# Consonant Recognition by Modular Construction of Large Phonemic Time-Delay Neural Networks

Alex Waibel
Carnegie-Mellon University
Pittsburgh, PA 15213,
ATR Interpreting Telephony Research Laboratories
Osaka, Japan

## Abstract

In this paper[1] we show that neural networks for speech recognition can be constructed in a modular fashion by exploiting the hidden structure of previously trained phonetic subcategory networks. The performance of resulting larger phonetic nets was found to be as good as the performance of the subcomponent nets by themselves. This approach avoids the excessive learning times that would be necessary to train larger networks and allows for incremental learning. Large time-delay neural networks constructed incrementally by applying these modular training techniques achieved a recognition performance of 96.0% for all consonants.

## 1. Introduction

Recently we have demonstrated that connectionist architectures capable of capturing some critical aspects of the dynamic nature of speech, can achieve superior recognition performance for difficult but small phonemic discrimination tasks such as discrimination of the voiced consonants B,D and G [Waibel 89, Waibel 88a]. Encouraged by these results we wanted to explore the question, how we might expand on these models to make them useful for the design of speech recognition systems. A problem that emerges as we attempt to apply neural network models to the full speech recognition problem is the problem of scaling. Simply extending neural networks to ever larger structures and retraining them as one monolithic net quickly exceeds the capabilities of the fastest and largest supercomputers. The search complexity of finding a good solutions in a huge space of possible network configurations also soon assumes unmanageable proportions. Moreover, having to decide on all possible classes for recognition ahead of time as well as collecting sufficient data to train such a large monolithic network is impractical to say the least. In an effort to extend our models from small recognition tasks to large scale speech recognition systems, we must therefore explore modularity and incremental learning as design strategies to break up a large learning task into smaller subtasks. Breaking up a large task into subtasks to be tackled by individual black boxes interconnected in ad hoc arrangements, on the other hand, would mean to abandon one of the most attractive aspects of connectionism: the ability to perform complex constraint satisfaction in a massively parallel and interconnected fashion, in view of an overall optimal performance goal. In this paper we demonstrate based on a set of experiments aimed at phoneme recognition that it is indeed possible to construct large neural networks incrementally by exploiting the hidden structure of smaller pretrained subcomponent

networks.

## 2. Small Phonemic Classes by Time-Delay Neural Networks

In our previous work, we have proposed a Time-Delay Neural Network architecture (as shown on the left of Fig.1 for B,D,G) as an approach to phoneme discrimination that achieves very high recognition scores [Waibel 89, Waibel 88a]. Its multilayer architecture, its shift-invariance and the time delayed connections of its units all contributed to its performance by allowing the net to develop complex, non-linear decision surfaces and insensitivity to misalignments and by incorporating contextual information into decision making (see [Waibel 89, Waibel 88a] for detailed analysis and discussion). It is trained by the back-propagation procedure [Rumelhart 86] using shared weights for different time shifted positions of the net [Waibel 89, Waibel 88a]. In spirit it has similarities to other models recently proposed [Watrous 88, Tank 87]. This network, however, had only been trained for the voiced stops B,D,G and we began our extensions by training similar networks for the other phonemic classes in our database.

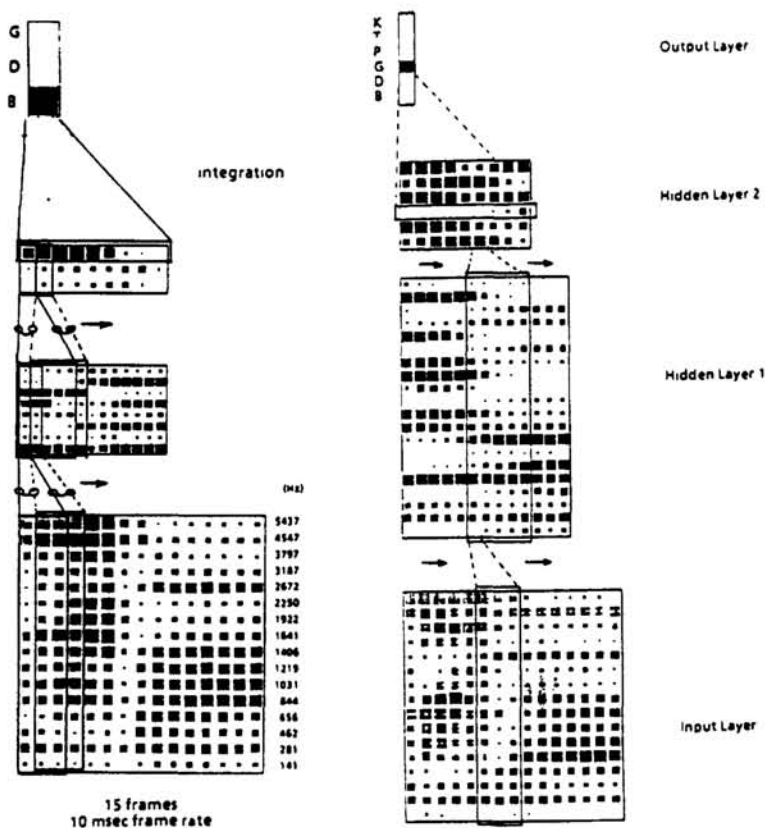

**Figure 1.** The TDNN architecture: BDG-net (left), BDGPTK-net (right)

All phoneme tokens in our experiments were extracted using phonetic handlabels from a large vocabulary database of 5240 common Japanese words. Each word in the database was spoken in isolation by one male native Japanese speaker. All utterances were recorded in a sound proof booth and digitized at a 12 kHz sampling rate. The database was then split into a training set and a testing set of 2620 utterances each. A 150 msec range around a phoneme boundary was excised for each phoneme token and 16 mel scale filterbank coefficients computed every 10 msec [Waibel 89, Waibel 88a]. The

preprocessed training and testing data was then used to train or to evaluate our TDNNs' performance for various phoneme classes. For each class, TDNNs with an architecture similar to the BDG-net in Fig.1 were trained. A total of seven nets aimed at the major coarse phonetic classes in Japanese were trained, including voiced stops B, D, G, voiceless stops P,T,K, the nasals M, N and syllabic nasals, fricatives S, SH, H and Z, affricates CH, TS, liquids and glides R, W, Y and finally the set of vowels A, I, U, E and O. Each of these nets was given between two and five phonemes to distinguish and the pertinent input data was presented for learning. Note, that each net was trained only within each respective coarse class and has no notion of phonemes from other classes yet. Evaluation of each net on test data within each of these subcategories revealed that an average rate of 98.8% can be achieved (see [Waibel 88b] for a more detailed tabulation of results).

## 3. Scaling TDNNs to Larger Phonemic Classes

We have seen that TDNNs achieve superior recognition performance on difficult but small recognition tasks. To train these networks substantial computational resources were needed. This raises the question of how our networks could be extended to encompass *all* phonemes or handle speech recognition in general. To shed light on this question of scaling, we consider first the problem of extending our networks from the task of voiced stop consonant recognition (hence the BDG-task) to the task of distinguishing among *all* stop consonants (the BDGPTK-task).

For a network aimed at the discrimination of the voiced stops (a BDG-net), approximately 6000 connections had to be trained over about 800 training tokens. An identical net (also with approximately 6000 connections[2]) can achieve discrimination among the voiceless stops ("P", "T" and "K"). To extend our networks to the recognition of *all* stops, i.e., the voiced *and* the unvoiced stops (B,D,G,P,T,K), a larger net is required. We have trained such a network for experimental purposes. To allow for the necessary number of features to develop we have given this net 20 units in the first hidden layer, 6 units in hidden layer 2 and 6 output units. On the right of Fig.1 we show this net in actual operation with a "G" presented at its input. Eventually a high performance network was obtained that achieves 98.3% correct recognition over a 1613-token BDGPTK-test database, but it took inordinate amounts of learning to arrive at the trained net (18 days on a 4 processor Alliant!). Although going from voiced stops to all stops is only a modest increase in task size, about 18,000 connections had to be trained. To make matters worse, not only the number of connections should be increased with task size, but in general the amount of training data required for good generalization of a larger net has to be increased as well. Naturally, there are practical limits to the size of a training database, and more training data translates into even more learning time. Learning is further complicated by the increased complexity of the higher dimensional weightspace in large nets as well as the limited precision of our simulators. Despite progress towards faster learning algorithms [Haffner 88, Fahlman 88], it is clear that we cannot hope for one single monolithic network to be trained within reasonable time as we

increase size to handle larger and larger tasks. Moreover, requiring that all classes be considered and samples of each class be presented during training, is undesirable for practical reasons as we contemplate the design of large neural systems. Alternative ways to modularly construct and incrementally train such large neural systems must therefore be explored.

## 3.1. Experiments with Modularity

Four experiments were performed to explore methodologies for constructing phonetic neural nets from smaller component subnets. As a task we used again stop consonant recognition (BDGPTK) although other tasks have recently been explored with similar success (BDG and MNsN) [Waibel 88c]. As in the previous section we used a large database of 5240 common Japanese words spoken in isolation from which the testing an training tokens for the voiced stops (the BDG-set) and for the voiceless stops (the PTK-set) was extracted.

Two separate TDNNs have been trained. On testing data the BDG-net used here performed 98.3% correct for the BDG-set and the PTK-net achieved 98.7% correct recognition for the PTK-set. As a first naive attempt we have now simply run a speech token from either set (i.e., B,D,G,P,T or K) through both a BDG-net and a PTK-net and selected the class with the *highest activation* from either net as the recognition result. As might have been expected (the component nets had only been trained for their respective classes), poor recognition performance (60.5%) resulted from the 6 class experiment. This is partially due to the inhibitory property of the TDNN that we have observed elsewhere [Waibel 89]. To combine the two networks more effectively, therefore, portions of the net had to be retrained.

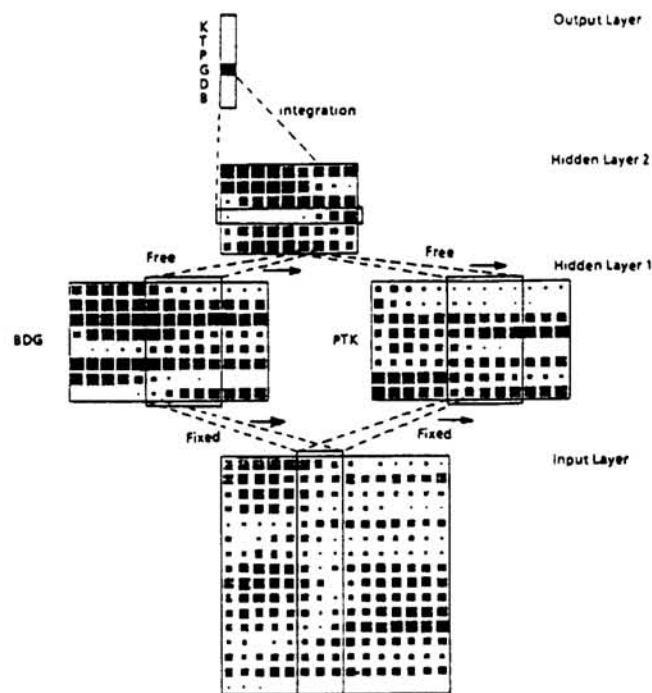

**Figure 2.** BDGPTK-net trained from hidden units from a BDG- and a PTK-net.

We start by assuming that the first hidden layer in either net already contains all the lower

level acoustic phonetic features we need for proper identification of the stops and freeze the connections from the input layer (the speech data) to the first hidden layer's 8 units in the BDG-net and the 8 units in the PTK-net. Back-propagation learning is then performed only on the connections between these 16 (= 2 X 8) units in hidden layer 1 and hidden layer 2 and between hidden layer 2 and the combined BDGPTK-net's output. This network is shown in Fig.2 with a "G" token presented as input. Only the higher layer connections had to be retrained (for about one day) in this case and the resulting network achieved a recognition performance of 98.1% over the testing data. Combination of the two subnets has therefore yielded a good net although a slight performance degradation compared to the subnets was observed. This degradation could be explained by the increased complexity of the task, but also by the inability of this net to develop lower level acoustic-phonetic features in hidden layer 1. Such features may in fact be needed for discrimination *between* the two stop classes, in addition to the within-class features.

In a third experiment, we therefore first train a separate TDNN to perform the voiced/unvoiced (V/UV) distinction between the BDG- and the PTK-task. The network has a very similar structure as the BDG-net, except that only four hidden units were used in hidden layer 1 and two in hidden layer 2 and at the output. This V/UV-net achieved better than 99% voiced/unvoiced classification on the test data and its hidden units developed in the process are now used as additional features for the BDGPTK-task. The connections from the input to the first hidden layer of the BDG-, the PTK- and the V/UV nets are frozen and only the connections that combine the 20 units in hidden layer 1 to the higher layers are retrained. Training of the V/UV-net and subsequent combination training took between one and two days. The resulting net was evaluated as before on our testing database and achieved a recognition score of 98.4% correct.

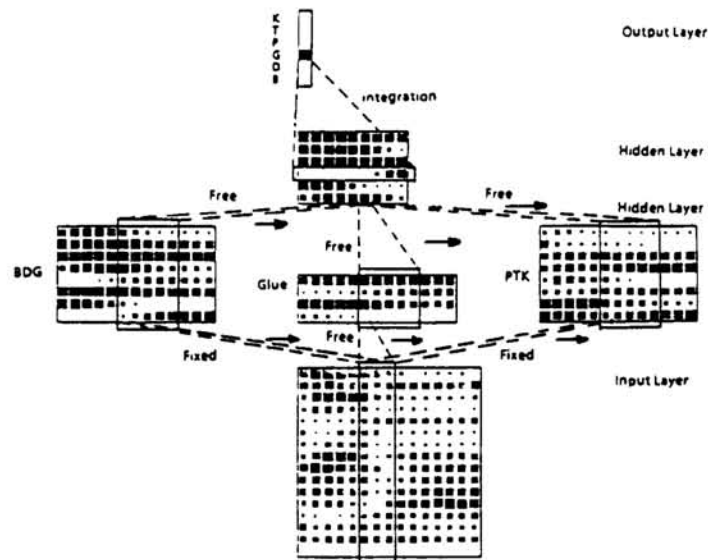

**Figure 3.** Combination of a BDG-net and a PTK-net using
4 additional units in hidden layer 1 as free "Connectionist Glue".

In the previous experiment, good results could be obtained by adding units that we *believed* to be the useful class distinctive features that were missing in our second experiment. In a fourth experiment, we have now examined an approach that allows for

the network to be free to discover *any* additional features that might be useful to merge the two component networks. In stead of previously training a class distinctive network, we now add four units to hidden layer 1, whose connections to the input are free to learn any missing discriminatory features to supplement the 16 frozen BDG and PTK features. We call these units the "*connectionist glue*" that we apply to merge two distinct networks into a new combined net. This network is shown in Fig.3. The hidden units of hidden layer 1 from the BDG-net are shown on the left and those from the PTK-net on the right. The connections from the moving input window to these units have been trained individually on BDG- and PTK-data, respectively, and -as before- remain fixed during combination learning. In the middle on hidden layer 1 we show the 4 free "Glue" units. Combination learning now searches for an optimal combination of the existing BDG- and PTK-features and also supplements these by learning additional interclass discriminatory features. Combination retraining with "glue" required a two day training run. Performance evaluation of this network over the BDGPTK test database yielded a recognition rate of 98.4%.

In addition to the techniques described so far, it may be useful to free *all* connections in a large modularly constructed network for an additional small amount of fine tuning. This has been done for the BDGPTK-net shown in Fig.3 yielding some additional performance improvements. Each iteration of the full network is indeed very slow, but convergence is reached after only few additional tuning iterations. The resulting network finally achieved (over testing data) a recognition score of 98.6%.

## 3.2. Steps for the Design of Large Scale Neural Nets

| Method | bdg | ptk | bdgptk |
|---|---|---|---|
| Individual TDNNs | 98.3 % | 98.7 % | |
| TDNN:Max. Activation | | | 60.5 % |
| Retrain BDGPTK | | | 98.3 % |
| Retrain Combined Higher Layers | | | 98.1 % |
| Retrain with V/UV-units | | | 98.4 % |
| Retrain with Glue | | | 98.4 % |
| All-Net Fine Tuning | | | 98.6 % |

**Table 3-1:** From BDG to BDGPTK; Modular Scaling Methods.

Table 3-1 summarizes the major results from our experiments. In the first row it shows the recognition performance of the two initial TDNNs trained individually to perform the BDG- and the PTK-tasks, respectively. Underneath, we show the results from the various experiments described in the previous section. The results indicate, that larger TDNNs can indeed be trained *incrementally*, without requiring excessive amounts of training and without loss in performance. The total incremental training time was between one third and one half of a full monolithically trained net and the resulting

networks appear to perform slightly better. Even more astonishingly, they appear to achieve performance as high as the subcomponent BDG- and PTK-nets alone. As a strategy for the efficient construction of larger networks we have found the following concepts to be extremely effective: *modular,incremental learning, class distinctive learning, connectionist glue, partial and selective learning and all-net fine tuning.*

## 4. Recognition of all Consonants
The incremental learning techniques explored so far can now be applied to the design of networks capable of recognizing all consonants.

### 4.1. Network Architecture

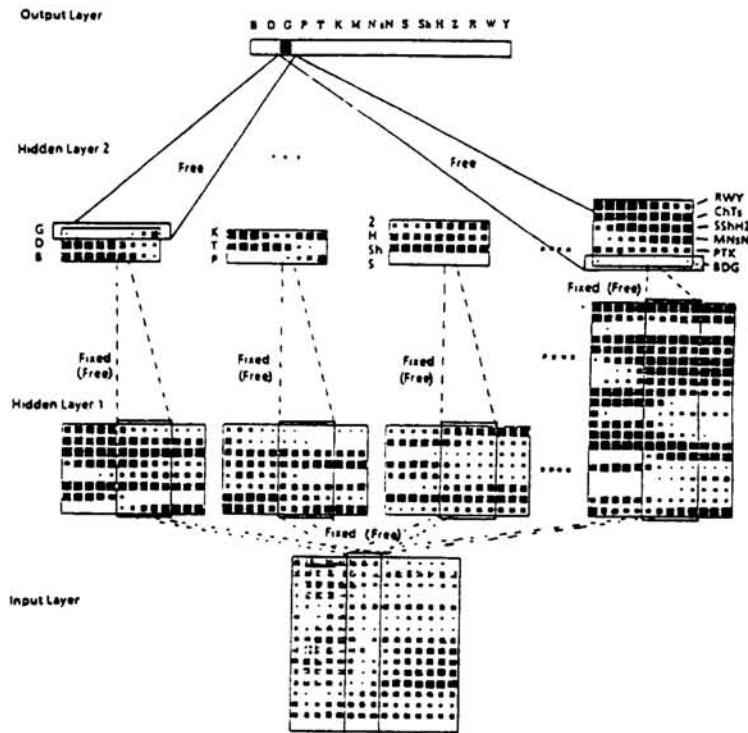

**Figure 4.** Modular Construction of an All Consonant Network

Our consonant TDNN (shown in Fig.4.1) was constructed modularly froth networks aimed at the consonant subcategories, i.e., the BDG-, PTK-, MNsN-, SShHZ-, TsCh- and the RWY-tasks. Each of these nets had been trained before to discriminate between the consonants *within* each class. Hidden layers 1 and 2 were then extracted from these nets, i.e. their weights copied and frozen in a new combined consonant TDNN. In addition, an interclass discrimination net was trained that distinguishes *between* the consonant subclasses and thus hopefully provides missing featural information for interclass discrimination much like the V/UV network described in the previous section. The structure of this network was very similar to other subcategory TDNNs, except that we have allowed for 20 units in hidden layer 1 and 6 hidden units (one for each coarse consonant class) in hidden layer 2. The weights leading into hidden layers 1 and 2 were then also copied from this interclass discrimination net into the consonant network and frozen. Three connections were then established to each of the 18 consonant output categories (B,D,G,P,T,K,M,N,sN,S, Sh,H,Z,Ch,Ts,R,W and Y): one to connect an output

unit with the appropriate *interclass* discrimination unit in hidden layer 2, one with the appropriate *intraclass* discrimination unit from hidden layer 2 of the corresponding subcategory net and one with the always activated threshold unit (not shown in Fig.4.1) The overall network architecture is shown in Fig.4.1 for the case of an incoming test token (e.g., a "G"). For simplicity, Fig.4.1 shows only the hidden layers from the BDG-,PTK,SShHZ- and the inter-class discrimination nets. At the output, only the two connections leading to the correctly activated "G"-output unit are shown. Units and connections pertaining to the other subcategories as well as connections leading to the 17 other output units are omitted for clarity in Fig.4.1. All free weights were initialized with small random weights and then trained.

## 4.2. Results

Consonants

| Task | Recognition Rate (%) |
|---|---|
| bdg | 98.6 |
| ptk | 98.7 |
| mnN | 96.6 |
| sshhz | 99.3 |
| chts | 100.0 |
| rwy | 99.9 |
| cons. class | 96.7 |
| All consonant TDNN | 95.0 |
| All-Net Fine Tuning | 95.9 |

**Table 4-1:** Consonant Recognition Performance Results.

Table 4.2 summarizes our results for the consonant recognition task. In the first 6 rows the recognition results (measured over the available test data in their respective sublasses) are given. The entry "cons.class" shows the performance of the interclass discrimination net in identifying the coarse phonemic subclass of an unknown token. 96.7% of all tokens were correctly categorized into one of the six consonant subclasses. After completion of combination learning the entire net was evaluated over 3061 consonant test tokens, and achieved a 95.0% recognition accuracy. All-net fine tuning was then performed by freeing up *all* connections in the network to allow for small additional adjustments in the interest of better overall performance. After completion of all-net fine tuning, the performance of the network then improved to 96.0% correct. To put these recognition results into perspective, we have compared these results with several other competing recognition techniques and found that our incrementally trained net compares favorably [Waibel 88b].

## 5. Conclusion

The serious problems associated with scaling smaller phonemic subcomponent networks to larger phonemic tasks are overcome by careful modular design. Modular design is achieved by several important strategies: *selective and incremental learning* of subcomponent tasks, *exploitation of previously learned hidden structure*, the application of *connectionist glue* or *class distinctive features* to allow for separate networks to "grow" together, *partial training* of portions of a larger net and finally, *all-net fine tuning* for making small additional adjustments in a large net. Our findings suggest, that judicious application of a number of connectionist design techniques could lead to the successful design of high performance large scale connectionist speech recognition systems.

## Footnotes

[1]An extended version of this paper will also appear in the Proceedings of the 1989 International Conference on Acoustics, Speech and Signal Processing. Copyright: IEEE. Reprinted with permission.

[2]Note, that these are connections over which a back-propagation pass is performed during each iteration. Since many of them share the same weights, only a small fraction (about 500) of them are actually free parameters.

## References

[Fahlman 88] Fahlman, S.E. *An Empirical Study of Learning Speed in Back-Propagation Networks.* Technical Report CMU-CS-88-162, Carnegie-Mellon University, June, 1988.

[Haffner 88] Haffner, P., Waibel, A. and Shikano, K. Fast Back-Propagation Learning Methods for Neural Networks in Speech. In *Proceedings of the Fall Meeting of the Acoustical Society of Japan.* October, 1988.

[Rumelhart 86] Rumelhart, D.E., Hinton, G.E. and Williams, R.J. Learning Internal Representations by Error Propagation. In McClelland, J.L. and Rumelhart, D.E. (editor), *Parallel Distributed Processing; Explorations in the Microstructure of Cognition*, chapter 8, pages 318-362. MIT Press, Cambridge, MA, 1986.

[Tank 87] Tank, D.W. and Hopfield, J.J. Neural Computation by Concentrating Information in Time. In *Proceedings National Academy of Sciences*, pages 1896-1900. April, 1987.

[Waibel 88a] Waibel, A., Hanazawa, T., Hinton, G., Shikano, K. and Lang K. Phoneme Recognition: Neural Networks vs. Hidden Markov Models. In *IEEE International Conference on Acoustics, Speech, and Signal Processing*, pages 8.S3.3. April, 1988.

[Waibel 88b] Waibel, A., Sawai, H. and Shikano, K. *Modularity and Scaling in Large Phonemic Neural Networks.* Technical Report TR-I-0034, ATR Interpreting Telephony Research Laboratories, July, 1988.

[Waibel 88c] Waibel, A. Connectionist Glue: Modular Design of Neural Speech Systems. In Touretzky, D.S., Hinton, G.E. and Sejnowski, T.J. (editors), *Proceedings of the 1988 Connectionist Models Summer School.* Morgan Kaufmann, 1988.

[Waibel 89] Waibel, A., Hanazawa, T., Hinton, G., Shikano, K. and Lang K. Phoneme Recognition Using Time-Delay Neural Networks. *IEEE, Transactions on Acoustics, Speech and Signal Processing* , March, 1989.

[Watrous 88] Watrous, R. *Speech Recognition Using Connectionist Networks.* PhD thesis, University of Pennsylvania, October, 1988.
